# Globally Optimal On-line Learning Rules

**Magnus Rattray\*and David Saad[†]**
Department of Computer Science & Applied Mathematics,
Aston University, Birmingham B4 7ET, UK.

## Abstract

We present a method for determining the globally optimal on-line learning rule for a soft committee machine under a statistical mechanics framework. This work complements previous results on locally optimal rules, where only the rate of change in generalization error was considered. We maximize the total reduction in generalization error over the whole learning process and show how the resulting rule can significantly outperform the locally optimal rule.

## 1 Introduction

We consider a learning scenario in which a feed-forward neural network model (the student) emulates an unknown mapping (the teacher), given a set of training examples produced by the teacher. The performance of the student network is typically measured by its generalization error, which is the expected error on an unseen example. The aim of training is to reduce the generalization error by adapting the student network's parameters appropriately.

A common form of training is on-line learning, where training patterns are presented sequentially and independently to the network at each learning step. This form of training can be beneficial in terms of both storage and computation time, especially for large systems. A frequently used on-line training method for networks with continuous nodes is that of stochastic gradient descent, since a differentiable error measure can be defined in this case. The stochasticity is a consequence of the training error being determined according to only the latest, randomly chosen, training example. This is to be contrasted with batch learning, where all the training examples would be used to determine the training error leading to a deterministic algorithm. Finding an effective algorithm for discrete networks is less straightforward as the error measure is not differentiable.

[†] saadd@aston.ac.uk

Often, it is possible to improve on the basic stochastic gradient descent algorithm and a number of modifications have been suggested in the literature. At late times one can use on-line estimates of second order information (the Hessian or its eigenvalues) to ensure asymptotically optimal performance (e.g., [1, 2]). A number of heuristics also exist which attempt to improve performance during the transient phase of learning (for a review, see [3]). However, these heuristics all require the careful setting of parameters which can be critical to their performance. Moreover, it would be desirable to have principled and theoretically well motivated algorithms which do not rely on heuristic arguments.

Statistical mechanics allows a compact description for a number of on-line learning scenarios in the limit of large input dimension, which we have recently employed to propose a method for determining globally optimal learning rates for on-line gradient descent [4]. This method will be generalized here to determine globally optimal on-line learning rules for both discrete and continuous machines. That is, rules which provide the maximum reduction in generalization error over the whole learning process. This provides a natural extension to work on locally optimal learning rules [5, 6], where only the rate of change in generalization error is optimized. In fact, for simple systems we sometimes find that the locally optimal rule is also globally optimal. However, global optimization seems to be rather important in more complex systems which are characterized by more degrees of freedom and often require broken permutation symmetries to learn perfectly. We will outline our general formalism and consider two simple and tractable learning scenarios to demonstrate the method.

It should be pointed out that the optimal rules derived here will often require knowledge of macroscopic properties related to the teacher's structure which would not be known in general. In this sense these rules do not provide practical algorithms as they stand, although some of the required macroscopic properties may be evaluated or estimated on the basis of data gathered as the learning progresses. In any case these rules provide an upper bound on the performance one could expect from a real algorithm and may be instrumental in designing practical training algorithms.

## 2    The statistical mechanics framework

For calculating the optimal on-line learning rule we employ the statistical mechanics description of the learning process. Under this framework, which may be employed for both smooth [7, 8] and discrete systems (e.g. [9]), the learning process is captured by a small number of self-averaging statistics whose trajectory is deterministic in the limit of large input dimension. In this analysis the relevant statistics are overlaps between weight vectors associated with different nodes of the student and teacher networks. The equations of motion for the evolution of these overlaps can be written in closed form and can be integrated numerically to describe the dynamics.

We will consider a general two-layer soft committee machine[1]. The desired teacher mapping is from an $N$-dimensional input space $\boldsymbol{\xi} \in \Re^N$ onto a scalar $\zeta \in \Re$, which the student models through a map $\sigma(\mathbf{J}, \boldsymbol{\xi}) = \sum_{i=1}^{K} g(\mathbf{J}_i \cdot \boldsymbol{\xi})$, where $g(x)$ is the activation function for the hidden layer, $\mathbf{J} \equiv \{\mathbf{J}_i\}_{1 \le i \le K}$ is the set of input-to-hidden adaptive weights for the $K$ hidden nodes and the hidden-to-output weights are set to 1. The activation of hidden node $i$ under presentation of the input pattern $\boldsymbol{\xi}^\mu$ is denoted $x_i^\mu = \mathbf{J}_i \cdot \boldsymbol{\xi}^\mu$.

Training examples are of the form $(\boldsymbol{\xi}^\mu, \zeta^\mu)$ where $\mu = 1, 2, \ldots, P$. The components of the independently drawn input vectors $\boldsymbol{\xi}^\mu$ are uncorrelated random variables with zero mean and unit variance. The corresponding output $\zeta^\mu$ is given by a deterministic teacher of a similar configuration to the student except for a possible difference in the number $M$ of hidden units and is of the form $\zeta^\mu = \sum_{n=1}^M g(\mathbf{B}_n \cdot \boldsymbol{\xi}^\mu)$, where $\mathbf{B} \equiv \{\mathbf{B}_n\}_{1 \leq n \leq M}$ is the set of input-to-hidden adaptive weights. The activation of hidden node $n$ under presentation of the input pattern $\boldsymbol{\xi}^\mu$ is denoted $y_n^\mu = \mathbf{B}_n \cdot \boldsymbol{\xi}^\mu$. We will use indices $i, j, k, l \ldots$ to refer to units in the student network and $n, m, \ldots$ for units in the teacher network. We will use the commonly used quadratic deviation $\epsilon(\mathbf{J}, \boldsymbol{\xi}) \equiv \frac{1}{2} \left[ \sigma(\mathbf{J}, \boldsymbol{\xi}) - \zeta \right]^2$, as the measure of disagreement between teacher and student. The most basic learning rule is to perform gradient descent on this quantity. Performance on a typical input defines the generalization error $\epsilon_g(\mathbf{J}) \equiv \langle \epsilon(\mathbf{J}, \boldsymbol{\xi}) \rangle_{\{\xi\}}$ through an average over all possible input vectors $\boldsymbol{\xi}$.

The general form of learning rule we will consider is,

$$\mathbf{J}_i^{\mu+1} = \mathbf{J}_i^\mu + \frac{1}{N} F_i^\mu(x^\mu, \zeta^\mu) \, \boldsymbol{\xi}^\mu \tag{1}$$

where $\mathbf{F} \equiv \{F_i\}$ depends only on the student activations and the teacher's output, and not on the teacher activations which are unobservable. Note that gradient descent on the error takes this general form, as does Hebbian learning and other training algorithms commonly used in discrete machines. The optimal $\mathbf{F}$ can also depend on the self-averaging statistics which describe the dynamics, since we know how they evolve in time. Some of these would not be available in a practical application, although for some simple cases the unobservable statistics can be deduced from observable quantities. This is therefore an idealization rather than a practical algorithm and provides a bound on the performance of a real algorithm.

The activations are distributed according to a multivariate Gaussian with covariances: $\langle x_i x_k \rangle = \mathbf{J}_i \cdot \mathbf{J}_k \equiv Q_{ik}$, $\langle x_i y_n \rangle = \mathbf{J}_i \cdot \mathbf{B}_n \equiv R_{in}$, and $\langle y_n y_m \rangle = \mathbf{B}_n \cdot \mathbf{B}_m \equiv T_{nm}$, measuring overlaps between student and teacher vectors. Angled brackets denote averages over input patterns. The covariance matrix completely describes the state of the system and in the limit of large $N$ we can write equations of motion for each macroscopic (the $T_{nm}$ are fixed and define the teacher):

$$\frac{dR_{in}}{d\alpha} = \langle F_i y_n \rangle \qquad \frac{dQ_{ik}}{d\alpha} = \langle F_i x_k + F_k x_i + F_i F_k \rangle, \tag{2}$$

where angled brackets now denote the averages over activations, replacing the averages over inputs, and $\alpha = \mu/N$ plays the role of a continuous time variable.

## 3 The globally optimal rule

Carrying out the averaging over input patterns one obtains an expression for the generalization error which depends exclusively on the overlaps $R, Q$ and $T$. Using the dependence of their dynamics (Eq. 2) on $\mathbf{F}$ one can easily calculate the locally optimal learning rule [5] by taking the functional derivative of $d\epsilon_g(\mathbf{F})/d\alpha$ to zero, looking for the rule that will maximize the reduction in generalization error at the present time step. This approach has been shown to be successful in some training scenarios but is likely to fail where the learning process is characterized by several phases of a different natures (e.g., multilayer networks).

The *globally optimal* learning rule is found by minimizing the total change in generalization error over a fixed time window,

$$\Delta\epsilon_g(\mathbf{F}) = \int_{\alpha_0}^{\alpha_1} \frac{d\epsilon_g}{d\alpha} \, d\alpha = \int_{\alpha_0}^{\alpha_1} \mathcal{L}(\mathbf{F}, \alpha) \, d\alpha. \tag{3}$$

This is a functional of the learning rule which we minimize by a variational approach.

First we can rewrite the integrand by expanding in terms of the equations of motion, each constrained by a Lagrange multiplier,

$$\mathcal{L}(\mathbf{F}, \alpha) = \sum_{in} \frac{\partial \epsilon_g}{\partial R_{in}} \frac{dR_{in}}{d\alpha} + \sum_{ik} \frac{\partial \epsilon_g}{\partial Q_{ik}} \frac{dQ_{ik}}{d\alpha} + \sum_{in} \lambda_{in} \left( \frac{dR_{in}}{d\alpha} - \langle F_i y_n \rangle \right)$$
$$+ \sum_{ik} \nu_{ik} \left( \frac{dQ_{ik}}{d\alpha} - \langle F_i x_k + F_k x_i + F_i F_k \rangle \right) . \qquad (4)$$

The expression for $\mathcal{L}$ still involve two multidimensional integrations over $\mathbf{x}$ and $\mathbf{y}$, so taking variations in $\mathbf{F}$, which may depend on $\mathbf{x}$ and $\zeta$ but not on $\mathbf{y}$, we find an expression for the optimal rule in terms of the Lagrange multipliers:

$$\mathbf{F} = -\mathbf{x} - \frac{1}{2}\nu^{-1}\lambda\,\overline{\mathbf{y}} \qquad (5)$$

where $\nu = [\nu_{ij}]$ and $\lambda = [\lambda_{in}]$. We define $\overline{\mathbf{y}}$ to be the teacher's expected field given the teacher's output and the student activations, which are observable quantities:

$$\overline{\mathbf{y}} = \int \mathbf{dy}\, \mathbf{y}\, p(\mathbf{y}|\mathbf{x}, \zeta) . \qquad (6)$$

Now taking variations in the overlaps w.r.t. the integral in Eq. (3) we find a set of differential equations for the Lagrange multipliers,

$$\frac{d\lambda_{km}}{d\alpha} = -\sum_{in} \lambda_{in} \frac{\partial \langle F_i y_n \rangle}{\partial R_{km}} - \sum_{ij} \nu_{ij} \frac{\partial \langle F_i x_j + F_j x_i + F_i F_j \rangle}{\partial R_{km}}$$
$$\frac{d\nu_{kl}}{d\alpha} = -\sum_{in} \lambda_{in} \frac{\partial \langle F_i y_n \rangle}{\partial Q_{kl}} - \sum_{ij} \nu_{ij} \frac{\partial \langle F_i x_j + F_j x_i + F_i F_j \rangle}{\partial Q_{kl}} , \qquad (7)$$

where $\mathbf{F}$ takes its optimal value defined in Eq. (5). The boundary conditions for the Lagrange multipliers are,

$$\lambda_{in}(\alpha_1) = \left.\frac{\partial \epsilon_g}{\partial R_{in}}\right|_{\alpha_1} \quad \text{and} \quad \nu_{ik}(\alpha_1) = \left.\frac{\partial \epsilon_g}{\partial Q_{ik}}\right|_{\alpha_1} , \qquad (8)$$

which are found by minimizing the rate of change in generalization error at $\alpha_1$, so that the globally optimal solution reduces to the locally optimal solution at this point, reflecting the fact that changes at $\alpha_1$ have no affect at other times.

If the above expressions do not yield an explicit formula for the optimal rule then the rule can be determined iteratively by gradient descent on the functional $\Delta\epsilon_g(\mathbf{F})$. To determine all the quantities necessary for this procedure requires that we first integrate the equations for the overlaps forward and then integrate the equations for the Lagrange multipliers backwards from the boundary conditions in Eq. (8).

## 4 Two tractable examples

In order to apply the above results we must be able to carry out the average in Eq. (6) and then in Eq. (7). These averages are also required to determine the locally optimal learning rule, so that the present method can be extended to any of the problems which have already been considered under the criteria of local optimality. Here we present two examples where the averages can be computed in closed form. The first problem we consider is a boolean perceptron learning a

linearly separable task where we retrieve the locally optimal rule [5]. The second problem is an over-realizable task, where a soft committee machine student learns a perceptron with a sigmoidal response. In this example the globally optimal rule significantly outperforms the locally optimal rule and exhibits a faster asymptotic decay.

**Boolean perceptron:** For the boolean perceptron we choose the activation function $g(x) = \text{sgn}(x)$ and both teacher and student have a single hidden node $(M = K = 1)$. The locally optimal rule was determined by Kinouchi and Caticha [5] and they supply the expected teacher field given the teacher output $\zeta = \text{sgn}(y)$ and the student field $x$ (we take the teacher length $T = 1$ without loss of generality),

$$\bar{y} = \frac{R}{Q}\left(x + \frac{\zeta\sqrt{\frac{2}{\pi}}\exp(-\frac{\gamma^2 x^2}{2})}{\gamma\,\text{erfc}\left(\frac{-\zeta x \gamma}{\sqrt{2}}\right)}\right) \quad \text{where} \quad \gamma = \frac{R}{\sqrt{Q^2 - R^2 Q}}\ . \tag{9}$$

Substituting this expression into the Lagrange multiplier dynamics in Eq. (7) shows that the ratio of $\lambda$ to $\nu$ is given by $\lambda/\nu = -2Q/R$, and Eq. (5) then returns the locally optimal value for the optimal rule:

$$F = \frac{\zeta\sqrt{\frac{2}{\pi}}\exp(-\frac{\gamma^2 x^2}{2})}{\gamma\,\text{erfc}\left(\frac{-\zeta x \gamma}{\sqrt{2}}\right)}\ . \tag{10}$$

This rule leads to modulated Hebbian learning and the resulting dynamics are discussed in [5]. We also find that the locally optimal rule is retrieved when the teacher is corrupted by output or weight noise [9].

**Soft committee machine learning a continuous perceptron:** In this example the teacher is an invertible perceptron $(M = 1)$ while the student is a soft committee machine with an arbitrary number $(K)$ of hidden nodes. We choose the activation function $g(x) = \text{erf}(x/\sqrt{2})$ for both the student and teacher since this allows the generalization error to be determined in closed form [7]. This is an example of an over-realizable task, since the student has greater complexity than is required to learn the teacher's mapping. The locally optimal rule for this scenario was determined recently [6].

Since the teacher is invertible, the expected teacher activation $\bar{y}$ is trivially equal to the true activation $y$. This leads to a particularly simple form for the dynamics (the $n$ suffix is dropped since there is only one teacher node),

$$\frac{dR_i}{d\alpha} = b_i T - R_i \qquad \frac{dQ_{ik}}{d\alpha} = b_i b_k T - Q_{ik}\ , \tag{11}$$

where we have defined $b_i = -\sum_j \nu_{ij}^{-1}\lambda_j/2$ and the optimal rule is given by $F_i = b_i y - x_i$. The Lagrange multiplier dynamics in Eq. (7) then show that the relative ratios of each Lagrange multiplier remain fixed over time, so that $b_i$ is determined by its boundary value (see Eq. (8)). It is straightforward to find solutions for long times, since the $b_i$ approach limiting values for very small generalization error (there are a number of possible solutions because of symmetries in the problem but any such solution will have the same performance for long times). For example, one possible solution is to have $b_1 = 1$ and $b_i = 0$ for all $i \neq 1$, which leads to an exponential decay of weights associated with all but a single node. This shows how the optimal performance is achieved when the complexity of the student matches that of the teacher.

Figure 1 shows results for a three node student learning a continuous perceptron. Clearly, the locally optimal rule performs poorly in comparison to the globally

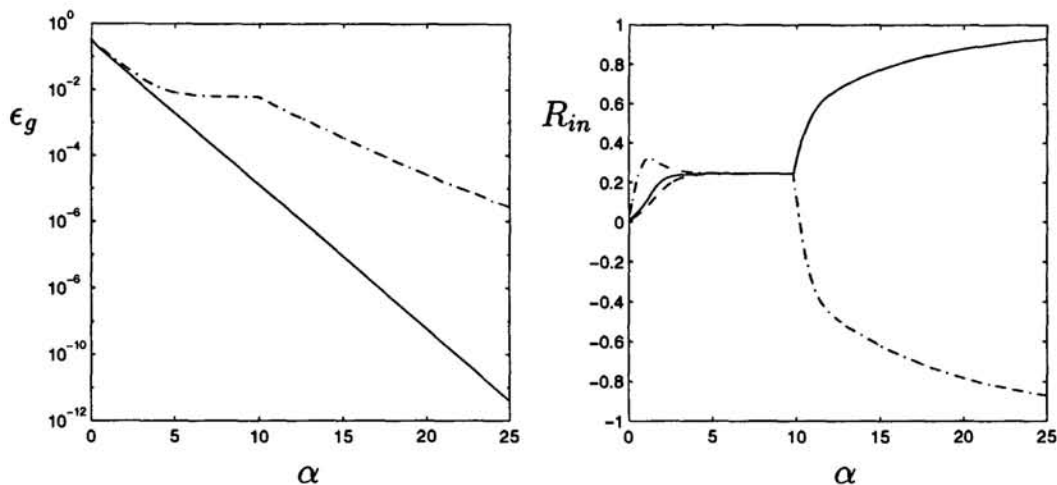

Figure 1: A three node soft committee machine student learns from an continuous perceptron teacher. The figure on the left shows a log plot of the generalization error for the globally optimal (solid line) and locally optimal (dashed line) algorithms. The figure on the right shows the student-teacher overlaps for the locally optimal rule, which exhibit a symmetric plateau before specialization occurs. The overlaps where initialized randomly and uniformly with $Q_{ii} \in [0, 0.5]$ and $R_i, Q_{i \neq j} \in [0, 10^{-6}]$.

optimal rule. In this example the globally optimal rule arrived at was one in which two nodes became correlated with the teacher while a third became anti-correlated, showing another possible variation on the optimal rule (we determined this rule iteratively by gradient descent in order to justify our general approach, although the observations above show how one can predict the final result for long times). The locally optimal rule gets caught in a symmetric plateau, characterized by a lack of differentiation between student vectors associated with different nodes, and also displays a slower asymptotic decay.

## 5 Conclusion and future work

We have presented a method for determining the optimal on-line learning rule for a soft committee machine under a statistical mechanics framework. This result complements previous work on locally optimal rules which sought only to optimize the rate of change in generalization error. In this work we considered the global optimization problem of minimizing the total change in generalization error over the whole learning process. We gave two simple examples for which the rule could be determined in closed form, for one of which, an over-realizable learning scenario, it was shown how the locally optimal rule performed poorly in comparison to the globally optimal rule. It is expected that more involved systems will show even greater difference in performance between local and global optimization and we are currently applying the method to more general teacher mappings. The main technical difficulty is in computing the expected teacher activation in Eq. (6) and this may require the use of approximate methods in some cases.

It would be interesting to compare the training dynamics obtained by the globally optimal rules to other approaches, heuristic and principled, aimed at incorporating information about the curvature of the error surface into the parameter modification rule. In particular we would like to examine rules which are known to be optimal *asymptotically* (e.g. [10]). Another important issue is whether one can apply these results to facilitate the design of a practical learning algorithm.

**Acknowledgement** This work was supported by the EPSRC grant GR/L19232.

## Footnotes

\* rattraym@aston.ac.uk

[1]The general result presented here also applies to the discrete committee machine, but we will limit our discussion to the soft-committee machine.

# References

[1] G. B. Orr and T. K. Leen in *Advances in Neural Information Processing Systems, vol 9,* eds M. C. Mozer, M. I. Jordan and T. Petsche (MIT Press, Cambridge MA, 1997) p 606.

[2] Y. LeCun, P. Y. Simard and B. Pearlmutter in *Advances in Neural Information Processing Systems, vol 5,* eds S. J. Hanson, J. D. Cowan and C. L. Giles (Morgan Kaufman, San Mateo, CA, 1993) p 156.

[3] C. M. Bishop, *Neural networks for pattern recognition,* (Oxford University Press, Oxford, 1995).

[4] D. Saad and M. Rattray, *Phys. Rev. Lett.* **79**, 2578 (1997).

[5] O. Kinouchi and N. Caticha *J. Phys. A* **25**, 6243 (1992).

[6] R. Vicente and N. Caticha *J. Phys. A* **30**, L599 (1997).

[7] D. Saad and S. A. Solla, *Phys. Rev. Lett.* **74**, 4337 (1995) and *Phys. Rev. E* **52** 4225 (1995).

[8] M. Biehl and H. Schwarze, *J. Phys. A* **28**, 643 (1995).

[9] M. Biehl, P. Riegler and M. Stechert, *Phys. Rev. E* **52**, R4624 (1995).

[10] S. Amari in *Advances in Neural Information Processing Systems, vol 9,* eds M. C. Mozer, M. I. Jordan and T. Petsche (MIT Press, Cambridge MA, 1997).